# From Online to Batch Learning with Cutoff-Averaging

## Abstract

We present *cutoff averaging*, a technique for converting any conservative online learning algorithm into a batch learning algorithm. Most online-to-batch conversion techniques work well with certain types of online learning algorithms and not with others, whereas cutoff averaging explicitly tries to adapt to the characteristics of the online algorithm being converted. An attractive property of our technique is that it preserves the efficiency of the original online algorithm, making it appropriate for large-scale learning problems. We provide a statistical analysis of our technique and back our theoretical claims with experimental results.

## 1 Introduction

*Batch learning* (also called *statistical learning*) and *online learning* are two different supervised machine-learning frameworks. In both frameworks, a learning problem is primarily defined by an instance space $\mathcal{X}$ and a label set $\mathcal{Y}$, and the goal is to assign labels from $\mathcal{Y}$ to instances in $\mathcal{X}$. In batch learning, we assume that there exists a probability distribution over the product space $\mathcal{X} \times \mathcal{Y}$, and that we have access to a training set drawn i.i.d. from this distribution. A batch learning algorithm uses the training set to generate an *output hypothesis*, which is a function that maps instances in $\mathcal{X}$ to labels in $\mathcal{Y}$. We expect a batch learning algorithm to *generalize*, in the sense that its output hypothesis should accurately predict the labels of previously unseen examples, which are sampled from the distribution.

On the other hand, in the online learning framework, we typically make no statistical assumptions regarding the origin of the data. An online learning algorithm receives a sequence of examples and processes these examples one-by-one. On each online-learning round, the algorithm receives an instance and predicts its label using an internal hypothesis, which it keeps in memory. Then, the algorithm receives the correct label corresponding to the instance, and uses the new instance-label pair to update and improve its internal hypothesis. There is no notion of statistical generalization, as the algorithm is only expected to accurately predict the labels of examples it receives as input. The sequence of internal hypotheses constructed by the online algorithm from round to round plays a central role in this paper, and we refer to this sequence as the *online hypothesis sequence*.

Online learning algorithms tend to be computationally efficient and easy to implement. However, many real-world problems fit more naturally in the batch learning framework. As a result, we are sometimes tempted to use online learning algorithms as if they were batch learning algorithms. A common way to do this is to present training examples one-by-one to the online algorithm, and use the last hypothesis constructed by the algorithm as the output hypothesis. We call this technique the *last-hypothesis* online-to-batch conversion technique. The appeal of this technique is that it maintains the computational efficiency of the original online algorithm. However, this heuristic technique generally comes with no theoretical guarantees, and the online algorithm's inherent disregard for out-of-sample performance makes it a risky practice.

In addition to the last-hypothesis heuristic, various principled techniques for converting online algorithms into batch algorithms have been proposed. Each of these techniques essentially wraps the online learning algorithm with an additional layer of instructions that endow it with the ability to generalize. One approach is to use the online algorithm to create the online hypothesis sequence, and then to choose a *single* good hypothesis from this sequence. For instance, the *longest survivor* technique [8] (originally called the pocket algorithm) chooses the hypothesis that survives the longest number of consecutive online rounds before it is replaced. The *validation* technique [12] uses a validation set to evaluate each online hypothesis and chooses the hypothesis with the best empirical performance. Improved versions of the validation technique are given in [2, 3], where the wasteful need for a separate validation set is resolved. All of these techniques follow the *single hypothesis* approach. We note in passing that a disadvantage of the various validation techniques [12, 2, 3] is that their running time scales quadratically with the number of examples. We typically turn to online algorithms for their efficiency, and often a quadratic running time can be problematic.

Another common online-to-batch conversion approach, which we call the *ensemble* approach, uses the online algorithm to construct the online hypothesis sequence, and combines the hypotheses in the sequence by taking a majority [7] or by averaging [2, Sec. 2.A]. When using linear hypotheses, averaging can be done on-the-fly, while the online algorithm is constructing the online hypothesis sequence. This preserves the computational efficiency of the online algorithm. Taking the majority or the average over a rich set of hypotheses promotes robustness and stability. Moreover, since we do not truly know the quality of each online hypothesis, building an ensemble allows us to hedge our bets, rather than committing to a single online hypothesis.

Sometimes the ensemble approach outperforms the single hypothesis approach, while other times we see the opposite behavior (see Sec. 4 and [9]). Ideally, we would like a conversion technique that enjoys the best of both worlds: when a single good online hypothesis can be clearly identified, it should be chosen as the output hypothesis, but when a good hypothesis cannot be identified, we should play it safe and construct an ensemble.

A first step in this direction was taken in [10, 5], where the conversion technique selectively chooses which subset of online hypotheses to include in the ensemble. For example, the *suffix averaging* conversion [5] sets the output hypothesis to be the average over a suffix of the online hypothesis sequence, where the suffix length is determined by minimizing a theoretical upper-bound on the generalization ability of the resulting hypothesis. One extreme of this approach is to include the entire online hypothesis sequence in the ensemble. The other extreme reduces to the last-hypothesis heuristic. By choosing the suffix that gives the best theoretical guarantee, suffix averaging automatically balances the trade-off between these two extremes. Regretfully, this technique suffers from a computational efficiency problem. Specifically, the suffix averaging technique only chooses the suffix length after the entire hypothesis sequence has been constructed. Therefore, it must store the entire sequence in memory before it constructs the output hypothesis, and its memory footprint grows linearly with training set size. This is in sharp contrast to the last-hypothesis heuristic, which uses no memory aside from the memory used by the online algorithm itself. When the training set is massive, storing the entire online hypothesis sequence in memory is impossible.

In this paper, we present and analyze a new conversion technique called *cutoff averaging*. Like suffix averaging, it attempts to enjoy the best of the single hypothesis approach and of the ensemble approach. One extreme of our technique reduces to the simple averaging conversion technique, while the other extreme reduces to the longest-survivor conversion technique. Like suffix averaging, we search for the sweet-spot between these two extremes by explicitly minimizing a tight theoretical generalization bound. The advantage of our technique is that much of it can be performed on-the-fly, as the online algorithm processes the data. The memory required by cutoff averaging scales with *square-root* the number of training examples in the worst case, and is far less in the typically case.

This paper is organized as follows. In Sec. 2 we formally present the background for our approach. In Sec. 3 we present the cutoff averaging technique and provide a statistical generalization analysis for it. Finally, we demonstrate the merits of our approach with a set of experiments in Sec. 4.

## 2 Preliminaries

Recall that $\mathcal{X}$ is an instance domain and that $\mathcal{Y}$ is a set of labels, and let $\mathcal{H}$ be a hypothesis class, where each $h \in \mathcal{H}$ is a mapping from $\mathcal{X}$ to $\mathcal{Y}$. For example, we may be faced with a confidence-rated binary classification problem, where $\mathcal{H}$ is the class of linear separators. In this case, $\mathcal{X}$ is a subset of the Euclidean space $\mathbb{R}^n$, $\mathcal{Y}$ is the real line, and each hypothesis in $\mathcal{H}$ is a linear function parametrized by a weight vector $\mathbf{w} \in \mathbb{R}^n$ and defined as $h(\mathbf{x}) = \langle \mathbf{w}, \mathbf{x} \rangle$. We interpret $\mathrm{sign}(h(\mathbf{x}))$ as the actual binary label predicted by $h$, and $|h(\mathbf{x})|$ as the degree of confidence in this prediction.

The quality of the predictions made by $h$ is measured using a loss function $\ell$. We use $\ell(h; (\mathbf{x}, y))$ to denote the penalty incurred for predicting the label $h(\mathbf{x})$ when the correct label is actually $y$. Returning to the example of linear separators, a common choice of loss function is the *zero-one loss*, which is simply the indicator function of prediction mistakes. Another popular loss function is the *hinge loss*, defined as

$$\ell(h; (\mathbf{x}, y)) \;=\; \left\{ \begin{array}{ll} 1 - y\langle \mathbf{w}, \mathbf{x} \rangle & \text{if } y\langle \mathbf{w}, \mathbf{x} \rangle \leq 1 \\ 0 & \text{otherwise} \end{array} \right. \quad .$$

As noted above, in batch learning we assume the existence of a probability distribution $\mathcal{D}$ over the product space $\mathcal{X} \times \mathcal{Y}$. The input of a batch learning algorithm is a training set, sampled from $\mathcal{D}^m$. The *risk* of a hypothesis $h$, denoted by $\ell(h; \mathcal{D})$, is defined as the expected loss incurred by $h$ over examples sampled from $\mathcal{D}$. Formally,

$$\ell(h; \mathcal{D}) \;=\; \mathbb{E}_{(X,Y) \sim \mathcal{D}} \left[ \ell(h; (X, Y)) \right] \;\;.$$

We can talk about the zero-one-risk or the hinge-loss-risk, depending on which loss function we choose to work with. The goal of a batch learning algorithm for the hypothesis class $\mathcal{H}$ and for the loss function $\ell$ is to find a hypothesis $h^\star \in \mathcal{H}$ whose risk is as close as possible to $\inf_{h \in \mathcal{H}} \ell(h; \mathcal{D})$.

In online learning, the labeled examples take the form of a sequence $S = \left( (\mathbf{x}_i, y_i) \right)_{i=1}^{m}$. We typically refrain from making any assumptions on the process that generates $S$; it could very well be a stochastic process but it doesn't have to be. The online algorithm observes the examples in the sequence one-by-one, and incrementally constructs the sequence of online hypotheses $(h_i)_{i=0}^{m}$, where each $h_i \in \mathcal{H}$. The first hypotheses, $h_0$, is a *default hypothesis*, which is defined in advance. Before round $t$ begins, the algorithm has already constructed the prefix $(h_i)_{i=0}^{t-1}$. At the beginning of round $t$, the algorithm observes $\mathbf{x}_t$ and makes the prediction $h_{t-1}(\mathbf{x}_t)$. Then, the correct label $y_t$ is revealed and the algorithm suffers a loss of $\ell(h_{t-1}; (\mathbf{x}_t, y_t))$. Finally, the algorithm uses the new example $(\mathbf{x}_t, y_t)$ to construct the next hypothesis $h_t$. The update rule used to construct $h_t$ is the main component of the online learning algorithm. In this paper, we make the simplifying assumption that the update rule is deterministic, and we note that our derivation can be extended to randomized update rules. Since $S$ is not necessarily generated by any distribution $\mathcal{D}$, we cannot define the risk of an online hypothesis. Instead, the performance of an online algorithm is measured using the game-theoretic notion of *regret*. The regret of an online algorithm is defined as

$$\frac{1}{m} \sum_{i=1}^{m} \ell(h_{i-1}; (\mathbf{x}_i, y_i)) \;-\; \min_{\hat{h} \in \mathcal{H}} \frac{1}{m} \sum_{i=1}^{m} \ell\left( \hat{h}; (\mathbf{x}_i, y_i) \right) \;\;. \tag{1}$$

In words, regret measures how much better the algorithm could have done by using the best fixed hypothesis in $\mathcal{H}$ on all $m$ rounds. The goal of an online learning algorithm is to minimize regret.

To make things more concrete, we focus on two online learning algorithms for binary classification. The first is the classic Perceptron algorithm [13] and the second is a *finite-horizon margin-based* variant of the Perceptron, which closely resembles algorithms given in [11, 4]. The term *finite-horizon* indicates that the algorithm knows the total length of the sequence of examples before observing any data. The term *margin-based* indicates that the algorithm is concerned with minimizing the hinge-loss, unlike the classic Perceptron, which deals directly with the zero-one loss. Pseudo-code for both algorithms is given in Fig. 1. We chose these two particular algorithms because they exhibit two extreme behaviors when converted into batch learning algorithms. Specifically, if we were to present the classic Perceptron with an example-sequence $S$ drawn i.i.d. from a distribution $\mathcal{D}$, we would typically see large fluctuations in the zero-one-risk of the various online hypotheses. (see Sec. 4). Due to these fluctuations, the ensemble approach suits the classic Perceptron very well,

| PERCEPTRON | FINITE-HORIZON MARGIN-BASED PERCEPTRON |
|---|---|
| input $S = \big((\mathbf{x}_i, y_i)\big)_{i=1}^{m}$ <br> set $\mathbf{w}_0 = (0, \ldots, 0)$ <br> for $i = 1, \ldots, m$ <br> $\quad$ receive $\mathbf{x}_i$, predict $\mathrm{sign}\langle \mathbf{w}_{i-1}, \mathbf{x}_i \rangle$ <br> $\quad$ receive $y_i \in \{-1, +1\}$ <br> $\quad$ if $\mathrm{sign}\big(\langle \mathbf{w}_{i-1}, \mathbf{x}_i \rangle\big) \neq y_i$ <br> $\qquad \mathbf{w}_i \leftarrow \mathbf{w}_{i-1} + y_i \mathbf{x}_i$ | input $S = \big((\mathbf{x}_i, y_i)\big)_{i=1}^{m}$ s.t. $\|\mathbf{x}_i\|_2 \leq R$ <br> set $\mathbf{w}_0 = (0, \ldots, 0)$ <br> for $i = 1, \ldots, m$ <br> $\quad$ receive $\mathbf{x}_i$, predict $\mathrm{sign}\langle \mathbf{w}_{i-1}, \mathbf{x}_i \rangle$ <br> $\quad$ receive $y_i \in \{-1, +1\}$ <br> $\quad$ if $\ell(\mathbf{w}_{i-1}; (\mathbf{x}_i, y_i)) > 0$ <br> $\qquad \mathbf{w}'_{i-1} \leftarrow \mathbf{w}_{i-1} + \frac{y_i \mathbf{x}_i}{\sqrt{m}R}$ <br> $\qquad \mathbf{w}_i \leftarrow \frac{\mathbf{w}'_{i-1}}{\|\mathbf{w}'_{i-1}\|_2}$ |

Figure 1: Two versions of the Perceptron algorithm.

and typically outperforms any single hypothesis approach. On the other hand, if we were to repeat this experiment with the margin-based Perceptron, using hinge-loss-risk, we would typically see a monotonic decrease in risk from round to round. A possible explanation for this is the similarity between the margin-based Perceptron and some incremental SVM solvers [14]. The last hypothesis constructed by the margin-based Perceptron is typically better than any average. This difference between the classic Perceptron and its margin-based variant was previously observed in [9]. Ideally, we would like a conversion technique that performs well in both cases.

From a theoretical standpoint, the purpose of an online-to-batch conversion technique is to turn an online learning algorithm with a regret bound into a batch learning algorithm with a risk bound. We state a regret bound for the margin-based Perceptron, so that we can demonstrate this idea in the next section.

**Theorem 1.** *Let $S = \big((\mathbf{x}_i, y_i)\big)_{i=1}^{m}$ be a sequence of examples such that $\mathbf{x}_i \in \mathbb{R}^n$ and $y \in \{-1, +1\}$ and let $\ell$ denote the hinge loss. Let $\mathcal{H}$ be the set of linear separators defined by weight vectors in the unit $L_2$ ball. Let $(h_i)_{i=0}^{m}$ be the online hypothesis sequence generated by the margin-based Perceptron (see Fig. 1) when it processes $S$. Then, for any $\hat{h} \in \mathcal{H}$,*

$$\frac{1}{m} \sum_{i=1}^{m} \ell\big(h_{i-1}; (\mathbf{x}_i, y_i)\big) \; - \; \frac{1}{m} \sum_{i=1}^{m} \ell\big(\hat{h}; (\mathbf{x}_i, y_i)\big) \; \leq \; \frac{R}{\sqrt{m}} \; .$$

The proof of Thm. 1 is not much different from other regret bounds for Perceptron-like algorithms; for completeness we give the proof in [1].

## 3 Cutoff Averaging

We now present the cutoff averaging conversion technique. This technique can be applied to any conservative online learning algorithm that uses a convex hypothesis class $\mathcal{H}$. A conservative algorithm is one that modifies its online hypotheses only on rounds where a positive loss is suffered. On rounds where no loss is suffered, the algorithm keeps its current hypothesis, and we say that the hypothesis *survived* the round. The *survival time* of each distinct online hypothesis is the number of consecutive rounds it survives before the algorithm suffers a loss and replaces it with a new hypothesis.

Like the conversion techniques mentioned in Sec. 1, we start by applying the online learning algorithm to an i.i.d. training set, and obtaining the online hypothesis sequence $(h_i)_{i=0}^{m-1}$. Let $k$ be an arbitrary non-negative integer, which we call the *cutoff parameter*. Ultimately, our technique will set $k$ automatically, but for the time-being, assume $k$ is a predefined constant. Let $\Theta \subseteq (h_i)_{i=0}^{m-1}$ be the set of distinct hypotheses whose survival time is greater than $k$. The cutoff averaging technique defines the output hypothesis $h^\star$ as a weighted average over the hypotheses in $\Theta$, where the weight of a hypothesis with survival time $s$ is proportional to $s - k$. Intuitively, each hypothesis must qualify for the ensemble, by suffering no loss for $k$ consecutive rounds. The cutoff parameter $k$ sets the bar for acceptance into the ensemble. Once a hypothesis is included in the ensemble, its weight is determined by the number of additional rounds it perseveres after qualifying.

We present a statistical analysis of the cutoff averaging technique. We use capital-letter notation throughout our analysis to emphasize that our input is stochastic and that we are essentially analyzing random variables. First, we represent the sequence of examples as a sequence of random variables $\big((X_i, Y_i)\big)_{i=1}^{m}$. Once this sequence is presented to the online algorithm, we obtain the online hypothesis sequence $(H_i)_{i=1}^{m}$, which is a sequence of random functions. Note that each random function $H_i$ is deterministically defined by the random variables $((X_j, Y_j))_{j=1}^{i}$. Therefore, the risk of $H_i$ is also a deterministic function of $((X_j, Y_j))_{j=1}^{i}$. Since $(X_{i+1}, Y_{i+1})$ is sampled from $\mathcal{D}$ independently of $((X_j, Y_j))_{j=1}^{i}$, we observe that

$$\ell(H_i; \mathcal{D}) \;=\; \mathbb{E}\big[\ell\big(H_i; (X_{i+1}, Y_{i+1})\big)\big|\big((X_j, Y_j)\big)_{j=1}^{i}\big] \;. \tag{2}$$

In words, the *risk* of the random function $H_i$ equals the conditional expectation of the *online loss* suffered on round $i+1$, conditioned on the random examples 1 through $i$. This simple observation relates statistical risk with online loss, and is the key to converting regret bounds into risk bounds.

Define the sequence of binary random variables $(B_i)_{i=0}^{m-1}$ as follows

$$B_i \;=\; \begin{cases} 1 & \text{if } i = 0 \quad \text{or} \quad \text{if } i \geq k \text{ and } H_{i-k} = H_{i-k+1} = \ldots = H_i \\ 0 & \text{otherwise} \end{cases} \;. \tag{3}$$

Now define the output hypothesis

$$H_k^{\star} \;=\; \Big(\sum_{i=0}^{m-1} B_i\Big)^{-1} \sum_{i=0}^{m-1} B_i H_i \;. \tag{4}$$

Note that we automatically include the default hypothesis $H_0$ in the definition of $H_k^{\star}$. This technical detail makes our analysis more elegant, and is otherwise irrelevant. Also note that setting $k = 0$ results in $B_i = 1$ for all $i$, and would reduce our conversion technique to the standard averaging conversion technique. At the other extreme, as $k$ increases, our technique approaches the longest survivor conversion technique.

The following theorem bounds the risk of $H_k^{\star}$ using the online loss suffered on rounds where $B_i = 1$. The theorem holds only when the loss function $\ell$ is convex in its first argument and bounded in $[0, C]$. Note that this is indeed the case for the margin-based Perceptron and the hinge loss function. Since the margin-based Perceptron enforces $\|\mathbf{w}_i\| \leq 1$, and assuming that $\|\mathbf{x}_i\| \leq R$, it follows from the Cauchy-Schwartz inequality that $\ell \in [0, R+1]$. If the loss function is not convex, the theorem does not hold, but note that we can still bound the average risk of the hypotheses in the ensemble.

**Theorem 2.** *Let $k$ be a non-negative constant and let $\ell$ be a convex loss function such that $\ell(h; (\mathbf{x}, y)) \in [0, C]$. An online algorithm is given $m \geq 4$ independent samples from $\mathcal{D}$ and constructs the online hypothesis sequence $(H_i)_{i=0}^{m}$. Define $B_i$ and $H_k^{\star}$ as above, let $L_i = B_{i-1}\ell\big(H_{i-1}; (X_i, Y_i)\big)$ for all $i$ and let $\bar{L} = (\sum B_i)^{-1} \sum L_i$. For any $\delta \in (0, 1)$, with probability at least $1 - \delta$, it holds that*

$$\ell(H_k^{\star}; \mathcal{D}) \;<\; \bar{L} + \sqrt{\frac{2C \ln(\frac{m}{\delta})\bar{L}}{\sum B_i}} + \frac{7C \ln(\frac{m}{\delta})}{\sum B_i} \;.$$

To prove the theorem, we require the following tail bound, which is a corollary of Freedman's tail bound for martingales [6], similar to [3, Proposition 2].

**Lemma 1.** *Let $(L_i)_{i=1}^{m}$ be a sequence of real-valued random variables and let $(Z_i)_{i=1}^{m}$ be a sequence of arbitrary random variables such that $L_i = \mathbb{E}[L_i|(Z_j)_{j=1}^{i}]$ and $L_i \in [0, C]$ for all $i$. Define $U_i = \mathbb{E}[L_i|(Z_j)_{j=1}^{i-1}]$ for all $i$, and define $\bar{L}_t = \sum_{i=1}^{t} L_i$ and $\bar{U}_t = \sum_{i=1}^{t} U_i$ for all $t$. For any $m \geq 4$ and for any $\delta \in (0, 1)$, with probability at least $1 - \delta$, it holds that*

$$\forall\, t \in \{1, \ldots, m\} \quad \bar{U}_t \;<\; \bar{L}_t + \sqrt{2C \ln(\tfrac{m}{\delta})\bar{L}_t} + 7C \ln(\tfrac{m}{\delta}) \;.$$

Due to space constraints, the proof of Lemma 1 is given in [1]. It can also be reverse-engineered from [3, Proposition 2]. Equipped with Lemma 1, we now prove Thm. 2.

*Proof of Thm. 2.* Define $U_i = \mathbb{E}[L_i|((X_j, Y_j))_{j=1}^{i-1}]$ for all $i \in \{1, \ldots, m\}$, and define $\bar{U} = \sum_{i=1}^{m} U_i$. Using Lemma 1, we have that, with probability at least $1 - \delta$

$$\bar{U} < \bar{L} + \sqrt{2C \ln(\tfrac{m}{\delta})\bar{L}} + 7C \ln(\tfrac{m}{\delta}) \ .$$

Now notice that, by definition,

$$U_i = \mathbb{E}\Big[B_{i-1}\ell\big(H_{i-1}; (X_i, Y_i)\big) \,\big|\, ((X_j, Y_j))_{j=1}^{i-1}\Big] \ .$$

Since $B_i$ is deterministically defined by $((X_j, Y_j))_{j=1}^{i-1}$, it can be taken outside of the conditional expectation above. Using the observation made in Eq. (2), we have $U_i = B_{i-1}\ell(H_{i-1}; \mathcal{D})$. Overall, we have shown that

$$\sum_{i=1}^{m} B_{i-1}\ell(H_{i-1}; \mathcal{D}) < \bar{L} + \sqrt{2C \ln(\tfrac{m}{\delta})\bar{L}} + 7C \ln(\tfrac{m}{\delta}) \ .$$

Using Jensen's inequality, the left-hand side above is at least $\big(\sum_{i=1}^{m} B_{i-1}\big)\ell(H_k^\star; \mathcal{D})$. $\qquad\square$

We can now complete the definition of the cutoff averaging technique. Note that by replacing $\delta$ with $\delta/m$ in Thm. 2 and by using the union bound, we can ensure that Thm. 2 holds uniformly for all $k \in \{0, \ldots, m - 1\}$ with probability at least $1 - \delta$. The cutoff averaging technique sets the output hypothesis $H^\star$ to be hypothesis in $\{H_0^\star, \ldots, H_{m-1}^\star\}$ for which Thm. 2 gives the smallest bound. In other words, $k$ is chosen automatically so as to balance the trade-off between the benefits of averaging and those of good empirical performance. If a small number of online hypotheses stand out with significantly long survival times, then our technique will favor a large $k$ and a sparse ensemble. On the other hand, if most of the online hypotheses have medium/short survival times, then our technique will favor small values of $k$ and a dense ensemble. Even if $\ell$ is not convex, minimizing the bound in Thm. 2 implicitly minimizes the average risk of the ensemble hypotheses.

If the online algorithm being converted has a regret bound, then the data dependent risk bound given by Thm. 2 can be turned into a data *independent* risk bound. A detailed derivation of such a bound exceeds the scope of this paper, and we just sketch the proof in the case of the margin-based Perceptron. It trivially holds that the risk of $H^\star$ is upper-bounded by the bound given in Thm. 2 for $k = 0$. When Thm. 2 is applied with $k = 0$, $\bar{L}$ simply becomes the average loss suffered by the online algorithm over the entire training set and $\sum B_i = m$. We can now use Thm. 1 to bound $\bar{L}$ by the average loss of any $\hat{h} \in \mathcal{H}$ on the sequence $((X_i, Y_i))_{i=1}^{m}$. Particularly, we can choose $\hat{h}$ to be the hypothesis with the smallest risk in $\mathcal{H}$, namely, $\hat{h} = \arg\min_{h \in \mathcal{H}} \ell(h; \mathcal{D})$. The final step is to bound the difference between $\frac{1}{m}\sum \ell(\hat{h}; (X_i, Y_i))$ and $\ell(\hat{h}; \mathcal{D})$, which can be done using any tail bound for sums of independent bounded random variables, such as Hoeffding's bound or Bernstein's bound. The result is that, with high probability, $\ell(H^\star; \mathcal{D}) \leq \min_{h \in \mathcal{H}} \ell(h; \mathcal{D}) + O(m^{-1/2})$. Similar derivations appear in [2, 3].

As mentioned in the introduction, our approach is similar to the suffix averaging conversion technique of [5], which also interpolates between an ensemble approach and a single hypothesis approach. However, the suffix conversion requires $\Omega(m)$ space, which is problematic when $m$ is large. In contrast, cutoff averaging requires only $O(\sqrt{m})$ space. Our technique cannot choose the optimal value of $k$ before the entire dataset has been processed, but nevertheless, it does not need to store the entire hypothesis sequence. Instead, it can group the online hypotheses based on their survival times, and stores only the average hypothesis in each group and the total loss in each group. By the time the entire dataset is processed, most of the work has already been done and calculating the optimal $k$ and the output hypothesis is straightforward. Using simple combinatorics, the maximal number of distinct survival times in a sequence of $m$ hypotheses is $O(\sqrt{m})$.

Finally, note that Lemma 1 is a Kolmogorov-type bound, namely, it holds uniformly for every prefix of the sequence of random variables. Therefore, Thm. 2 actually holds simultaneously for every prefix of the training set. Since our conversion is mostly calculated on-the-fly, in parallel with the online rounds, we can easily construct intermediate output hypotheses, before the online algorithm has a chance to process the entire dataset. Thanks to the Kolmorogorv-type bound, the risk bounds for all of these hypotheses all hold simultaneously. We can monitor how the risk bound changes as the number of examples increases, and perhaps even use the bound to define an early stopping criterion for the training algorithm. Specifically, we could stop processing examples when the risk bound becomes lower than a predefined threshold.

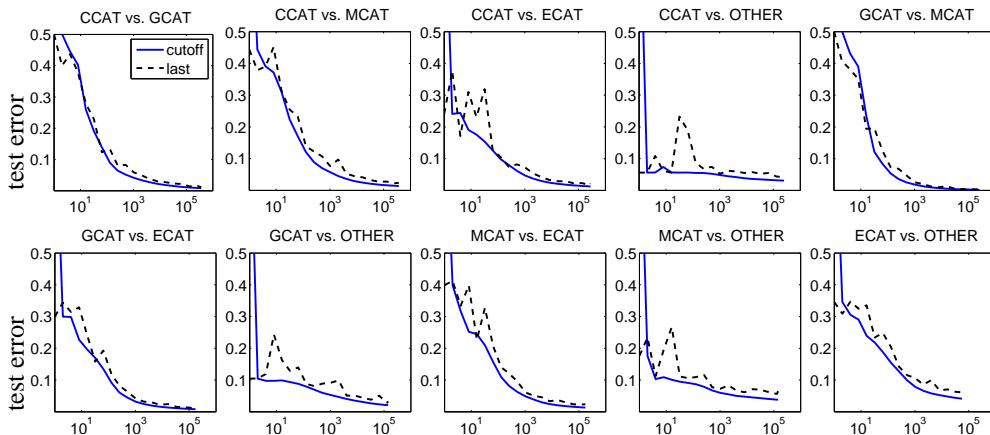

Figure 2: Test error (zero-one-loss) of *last-hypothesis* and *cutoff averaging*, each applied to the standard Perceptron, on ten binary classification problems from RCV1. The x-axis represents training set size, and is given in log-scale. Each plot represents the average over 10 random train-test splits.

## 4  Experiments and Conclusions

We conducted experiments using *Reuters Corpus Vol. 1* (RCV1), a collection of over 800K news articles collected from the Reuters news wire. An average article in the corpus contains 240 words, and the entire corpus contains over half a million distinct tokens (not including numbers and dates). Each article in the corpus is associated with one or more *high-level categories*, which are: Corporate/Industrial (CCAT), Economics (ECAT), Government/Social (GCAT), Markets (MCAT), and Other (OTHER). About $20\%$ of the articles in the corpus are associated with more than one high-level category. After discarding this $20\%$, we are left with over 600K documents, each with a single high-level label. Each pair of high-level labels defines the binary classification problem of distinguishing between articles of the two categories, for a total of ten different problems. Each problem has different characteristics, due to the different number of articles and the varying degree of homogeneity in each category.

Each article was mapped to a feature vector using a logarithmic bag-of-words representation. Namely, the length of each vector equals the number of distinct tokens in the corpus, and each coordinate in the vector represents one of these tokens. If a token appears $s$ times in a given article, the respective coordinate in the feature vector equals $\log_2(1 + s)$.

We applied the cutoff averaging technique to the classic Perceptron and to the margin-based Perceptron. We repeated each of our experiments ten times, each time taking a new random split of the data into a training set ($80\%$) and a test set ($20\%$), and randomly ordering the training set. We trained each algorithm on each dataset in an incremental manner, namely, we started by training the algorithm using a short prefix of the training sequence, and gradually increased the training set size. We paused training at regular intervals, computed the output hypothesis so far, and calculated its test loss. This gives us an idea of what would happen on smaller training sets.

Fig. 2 shows the test zero-one loss attained when our technique is applied to the classic Perceptron algorithm. It also shows the test zero-one loss of the last-hypothesis conversion technique. Clearly, the test loss of the last hypothesis is very unstable, even after averaging over 10 repetitions. In some cases, adding training data actually deteriorates the performance of the last hypothesis. If we decide to use the last hypothesis technique, our training set size could happen to be such that we end up with a bad output hypothesis. On the other hand, the cutoff averaging hypothesis is accurate, stable and consistent. The performance of the simple averaging conversion technique is not plotted in Fig. 2, but we note that it was only slightly worse than the performance of cutoff averaging. When using the classic Perceptron, any form of averaging is beneficial, and our technique successfully identifies this.

Fig. 3 shows the test hinge loss of cutoff averaging, last-hypothesis, and simple averaging, when applied to the margin-based Perceptron. In this case, the last hypothesis performs remarkably well

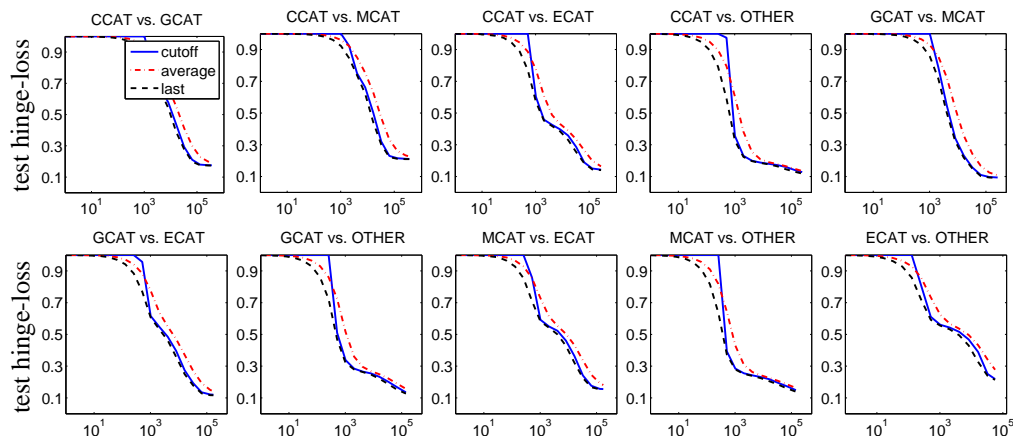

Figure 3: Test hinge-loss of *last-hypothesis*, *averaging*, and *cutoff averaging*, each applied to the finite-horizon margin-based Perceptron, on ten binary classification problems from RCV1. The x-axis represents training set size and each plot represents the average over 10 random train-test splits.

and the simple averaging conversion technique is significantly inferior for all training set sizes. Within 1000 online rounds (0.1% of the data), the cutoff averaging technique catches up to the last hypothesis and performs comparably well from then on. Our technique's poor performance on the first 0.1% of the data is expected, since the tail bounds we rely on are meaningless with so few examples. Once the tail bounds become tight enough, our technique essentially identifies that there is no benefit in constructing a diverse ensemble, and assigns all of the weight to a short suffix of the online hypothesis sequence.

We conclude that there are cases where the single-hypothesis approach is called for and there are cases where an ensemble approach should be used. If we are fortunate enough to know which case applies, we can simply choose the right approach. However, if we are after a generic solution that performs well in both cases, we need a conversion technique that automatically balances the trade-off between these two extremes. Suffix averaging [5] and cutoff averaging are two such techniques, with cutoff averaging having a significant computational advantage.

## References

[1] Anonimous. Technical appendix submitted with this manuscript, 2008.

[2] N. Cesa-Bianchi, A. Conconi, and C. Gentile. On the generalization ability of online learning algorithms. *IEEE Transactions on Information Theory*, 50(9):2050–2057, September 2004.

[3] N. Cesa-Bianchi and C. Gentile. Improved risk bounds for online algorithms. *NIPS 19*, 2006.

[4] O. Dekel, S. Shalev-Shwartz, and Y. Singer. The Forgetron: A kernel-based perceptron on a budget. *SIAM Journal on Computing*, 37:1342–1372, 2008.

[5] O. Dekel and Y. Singer. Data-driven online to batch conversions. *NIPS 18*, 2006.

[6] D. A. Freedman. On tail probabilities for martingales. *Annals of Prob.*, 3(1):100–118, 1975.

[7] Y. Freund and R. E. Schapire. Large margin classification using the perceptron algorithm. *Machine Learning*, 37(3):277–296, 1999.

[8] S. I. Gallant. Optimal linear discriminants. *Proc. of ICPR 8*, pages 849–852. IEEE, 1986.

[9] R. Khardon and G. Wachman. Noise tolerant variants of the perceptron algorithm. *Journal of Machine Learning Research*, 8:227–248, 2007.

[10] Y. Li. Selective voting for perceptron-like learning. *Proc. of ICML 17*, pages 559–566, 2000.

[11] Y. Li, H. Zaragoza, R. He, J. ShaweTaylor, and J. Kandola. The perceptron algorithm with uneven margins. *Proc. of ICML 19*, pages 379–386, 2002.

[12] N. Littlestone. From online to batch learning. *Proc. of COLT 2*, pages 269–284, 1989.

[13] F. Rosenblatt. The perceptron: A probabilistic model for information storage and organization in the brain. *Psychological Review*, 65:386–407, 1958.

[14] T. Zhang. Solving large scale linear prediction problems using stochastic gradient descent algorithms. *Proc. of ICML 21*, 2004.

